# Constructive Algorithms for Hierarchical Mixtures of Experts

**S.R.Waterhouse**          **A.J.Robinson**
Cambridge University Engineering Department,
Trumpington St., Cambridge, CB2 1PZ, England.
Tel: [+44] 1223 332754, Fax: [+44] 1223 332662,
Email: srw1001, ajr @eng.cam.ac.uk

## Abstract

We present two additions to the hierarchical mixture of experts (HME) architecture. By applying a likelihood splitting criteria to each expert in the HME we "grow" the tree adaptively during training. Secondly, by considering only the most probable path through the tree we may "prune" branches away, either temporarily, or permanently if they become redundant. We demonstrate results for the growing and path pruning algorithms which show significant speed ups and more efficient use of parameters over the standard fixed structure in discriminating between two interlocking spirals and classifying 8-bit parity patterns.

## INTRODUCTION

The HME (Jordan & Jacobs 1994) is a tree structured network whose terminal nodes are simple function approximators in the case of regression or classifiers in the case of classification. The outputs of the terminal nodes or experts are recursively combined upwards towards the root node, to form the overall output of the network, by "gates" which are situated at the non-terminal nodes.

The HME has clear similarities with tree based statistical methods such as Classification and Regression Trees (CART) (Breiman, Friedman, Olshen & Stone 1984). We may consider the gate as replacing the set of "questions" which are asked at each branch of CART. From this analogy, we may consider the application of the splitting rules used to build CART. We start with a simple tree consisting of two experts and one gate. After partially training this simple tree we apply the splitting criterion to each terminal node. This evaluates the log-likelihood increase by splitting each expert into two experts and a gate. The split which yields the best increase in log-likelihood is then added permanently to the tree. This process of training followed by growing continues until the desired modelling power is reached.

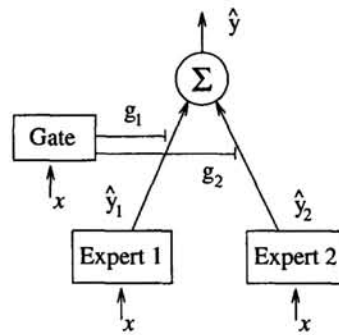

Figure 1: A simple mixture of experts.

This approach is reminiscent of Cascade Correlation (Fahlman & Lebiere 1990) in which new hidden nodes are added to a multi-layer perceptron and trained while the rest of the network is kept fixed.

The HME also has similarities with model merging techniques such as stacked regression (Wolpert 1993), in which explicit partitions of the training set are combined. However the HME differs from model merging in that each expert considers the whole input space in forming its output. Whilst this allows the network more flexibility since each gate may implicitly partition the whole input space in a "soft" manner, it leads to unnecessarily long computation in the case of near optimally trained models. At any one time only a few paths through a large network may have high probability. In order to overcome this drawback, we introduce the idea of "path pruning" which considers only those paths from the root node which have probability greater than a certain threshold.

## CLASSIFICATION USING HIERARCHICAL MIXTURES OF EXPERTS

The mixture of experts, shown in Figure 1, consists of a set of "experts" which perform local function approximation. The expert outputs are combined by a gate to form the overall output. In the hierarchical case, the experts are themselves mixtures of further experts, thus extending the architecture in a tree structured fashion. Each terminal node or "expert" may take on a variety of forms, depending on the application. In the case of multi-way classification, each expert outputs a vector $\hat{y}_j$ in which element $m$ is the conditional probability of class $m$ ($m = 1 \dots M$) which is computed using the softmax function:

$$P(c_m|x^{(n)}, \mathbf{w}_j) = \exp(\mathbf{w}_{m,j}^T x^{(n)}) \left/ \sum_{k=1}^{K} \exp(\mathbf{w}_{m,k}^T x^{(n)}) \right.$$

where $\mathbf{w}_j = [\mathbf{w}_{1j} \ \mathbf{w}_{2j} \ \dots \ \mathbf{w}_{Mj}]$ is the parameter matrix for expert $j$ and $c_i$ denotes class $i$.

The outputs of the experts are combined using a "gate" which sits at the non-terminal nodes. The gate outputs are estimates of the conditional probability of selecting the daughters of the non-terminal node given the input and the path taken to that node from the root node. This is once again computed using the softmax function:

$$P(z_j|x^{(n)}, \xi) = \exp(\xi_j^T x^{(n)}) \left/ \sum_{i=1}^{J} \exp(\xi_i^T x^{(n)}) \right.$$

where $\xi = [\xi_1 \ \xi_2 \ \dots \ \xi_J]$ is the parameter matrix for the gate, and $z_j$ denotes expert $j$.

The overall output is given by a probabilistic mixture in which the gate outputs are the mixture weights and the expert outputs are the mixture components. The probability of class $m$ is then given by:

$$P(c_m|x^{(n)}, \Theta) = \sum_{i=1}^{J} P(z_i|x^{(n)}, \xi) P(c_m|x^{(n)}, w_i).$$

A straightforward extension of this model also gives us the conditional probability $h_j^{(n)}$ of selecting expert $j$ given input $x^{(n)}$ *and* correct class $c_k$,

$$h_j^{(n)} \equiv P(z_j|c_k, x^{(n)}, w_j) = P(z_j|x^{(n)}, \xi) P(c_k|x^{(n)}, w_j) \bigg/ \sum_{i=1}^{J} P(z_i|x^{(n)}, \xi) P(c_k|x^{(n)}, w_i)$$

In order to train the HME to perform classification we maximise the log likelihood $L = \sum_{n=1}^{N} \sum_{m=1}^{M} t_m^{(n)} \log P(c_m|x^{(n)}, \Theta)$, where the variable $t_m^{(n)}$ is one if $m$ is the correct class at exemplar $(n)$ and zero otherwise. This is done via the expectation maximisation (EM) algorithm of Dempster, Laird & Rubin (1977), as described by Jordan & Jacobs (1994).

## TREE GROWING

The standard HME differs from most tree based statistical models in that its architecture is fixed. By relaxing this constraint and allowing the tree to grow, we achieve a greater degree of flexibility in the network. Following the work on CART we start with a simple tree, for instance with two experts and one gate which we train for a small number of cycles. Given this semi-trained network, we then make a set of candidate splits $\{S_i\}$ of terminal nodes $\{z_i\}$. Each split involves replacing an expert $z_i$ with a pair of new experts $\{z_{ij}\}_{j=1}^{2}$ and a gate, as shown in Figure 2.

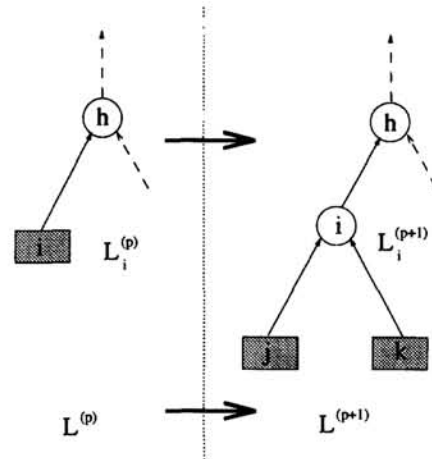

Figure 2: Making a candidate split of a terminal node.

We wish to select eventually only the "best" split $\tilde{S}$ out of these candidate splits. Let us define the best split as being that which maximises the increase in overall log-likelihood due to the split, $\Delta L = L^{(p+1)} - L^{(p)}$ where $L^{(p)}$ is the likelihood at the $p^{th}$ *generation* of the tree. If we make the constraint that all the parameters of the tree remain fixed apart from the parameters of the new split whenever a candidate split is made, then the maximisation is simplified into a dependency on the increases in the local likelihoods $\{L_i\}$ of the nodes $\{z_i\}$. We thus constrain the tree growing process to be localised such that we find the node which gains the most by being split.

$$\max_i \Delta L(S_i) \equiv \max_i \Delta L_i = \max_i (L_i^{(p+1)} - L_i^{(p)})$$

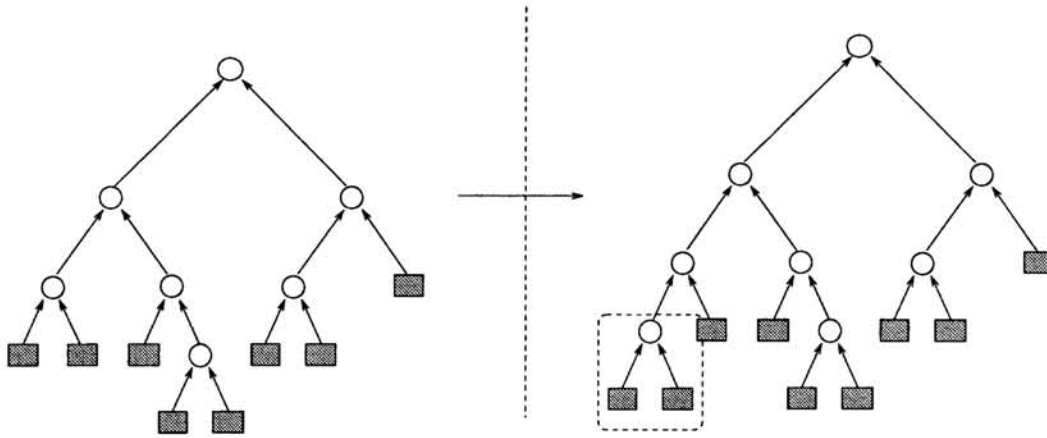

Figure 3: Growing the HME. This figure shows the addition of a pair of experts to the partially grown tree.

$$\text{where} \qquad L_i^{(p)} = \sum_n \sum_m t_m^{(n)} \log P(c_m | x^{(n)}, z_i, \mathbf{w}_i)$$

$$L_i^{(p+1)} = \sum_n \sum_m t_m^{(n)} \log \sum_j P(z_{ij} | x^{(n)}, \xi_i, z_i) P(c_m | x^{(n)}, z_{ij}, \mathbf{w}_{ij})$$

This splitting rule is similar in form to the CART splitting criterion which uses maximisation of the entropy of the node split, equivalent to our local increase in log-likelihood.

The final growing algorithm starts with a tree of generation $p$ and firstly fixes the parameters of all non-terminal nodes. All terminal nodes are then split into two experts and a gate. A split is only made if the sum of posterior probabilities $\sum_n h_i^{(n)}$, as described (1), at the node is greater than a small threshold. This prevents splits being made on nodes which have very little data assigned to them. In order to break symmetry, the new experts of a split are initialised by adding small random noise to the original expert parameters. The gate parameters are set to small random weights. For each node $i$, we then evaluate $\Delta L_i$ by training the tree using the standard EM method. Since all non-terminal node parameters are fixed the only changes to the log-likelihood are due the new splits. Since the parameters of each split are thus independent of one another, all splits can be trained at once, removing the need to train multiple trees separately.

After each split has been evaluated, the best split is chosen. This split is kept and all other splits are discarded. The original tree structure is then recovered except for the additional winning split, as shown in Figure 3. The new tree, of generation $p + 1$ is then trained as usual using EM. At present the decision on when to add a new split to the tree is fairly straightforward: a candidate split is made after training the fixed tree for a set number of iterations. An alternative scheme we have investigated is to make a split when the overall log-likelihood of the fixed tree has not increased for a set number of cycles. In addition, splits are rejected if they add too little to the local log-likelihood.

Although we have not discussed the issue of over-fitting in this paper, a number of techniques to prevent over-fitting can be used in the HME. The most simple technique, akin to those used in CART, involves growing a large tree and successively removing nodes from the tree until the performance on a cross validation set reaches an optimum. Alternatively the Bayesian techniques of Waterhouse, MacKay & Robinson (1995) could be applied.

**Tree growing simulations**

This algorithm was used to solve the 8-bit parity classification task. We compared the growing algorithm to a fixed HME with depth of 4 and binary branches. As can be seen in Figures 4(a) and (b), the factorisation enabled by the growing algorithm significantly speeds up computation over the standard fixed structure. The final tree shape obtained is shown in Figure 4(c). We showed in an earlier paper (Waterhouse & Robinson 1994) that the XOR problem may be solved using at least 2 experts and a gate. The 8 bit parity problem is therefore being solved by a series of XOR classifiers, each gated by its parent node, which is an intuitively appealing form with an efficient use of parameters.

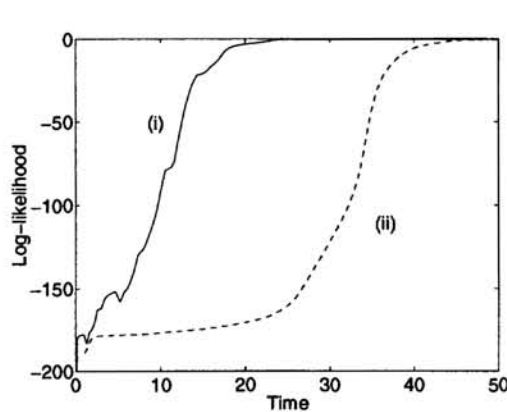

(a) Evolution of log-likelihood vs. time in CPU seconds.

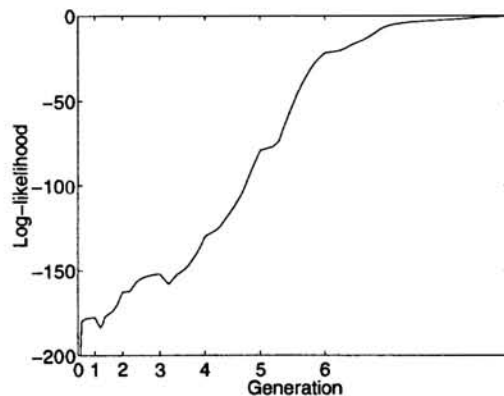

(b) Evolution of log-likelihood for (i) vs generations of tree.

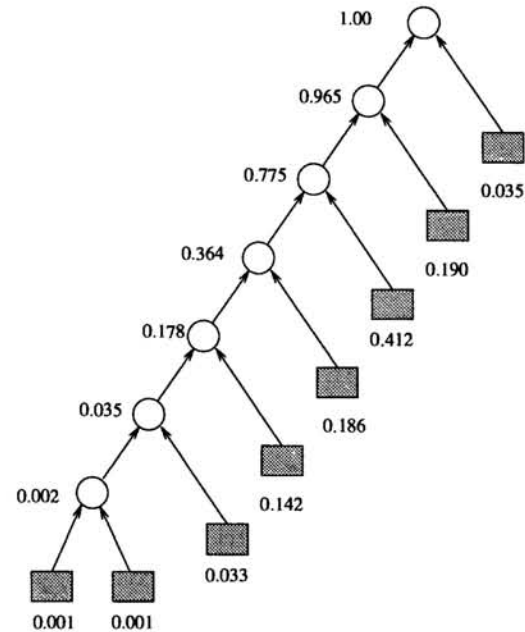

(c) Final tree structure obtained from (i), showing utilisation $U_i$ of each node where $U_i = \sum_n P(z_i, R_i | x^{(n)}) / N$, and $R_i$ is the path taken from the root node to node $i$.

Figure 4: HME GROWING ON THE 8 BIT PARITY PROBLEM;(i) growing HME with 6 generations; (ii) 4 deep binary branching HME (no growing).

**PATH PRUNING**

If we consider the HME to be a good model for the data generation process, the case for path pruning becomes clear. In a tree with sufficient depth to model the

underlying sub-processes producing each data point, we would expect the activation of each expert to tend to binary values such that only one expert is selected at each time exemplar.

The path pruning scheme is depicted in Figure 5. The pruning scheme utilises the "activation" of each node at each exemplar. The activation is defined as the product of node probabilities along a path from the root node to the current node, $J_i^{(n)} = \sum_i \log P(z_i|R_i, x^{(n)})$, where $R_i$ is the path taken to node $i$ from the root node. If $J_l^{(n)}$ for node $l$ at exemplar $n$ falls below a threshold value, $f_t$, then we ignore the subtree $S_l$ and we backtrack up to the parent node of $l$.

During training this involves not accumulating the statistics of the subtree $S_l$; during evaluation it involves setting the output of subtree $S_l$ to zero. In addition to this path pruning scheme we can use the activation of the nodes to do more permanent pruning. If the overall utilisation $U_i = \sum_n P(z_i, R_i|x^{(n)})/N$ of a node falls below a small threshold, then a node is pruned completely from the tree. The sister subtrees of the removed node then subsume their parent nodes. This process is used solely to improve computational efficiency in this paper, although conceivably it could be used as a regularisation method, akin to the brain surgery techniques of Cun, Denker & Solla (1990). In such a scheme, however, a more useful measure of node utilisation would be the *effective* number of parameters (Moody 1992).

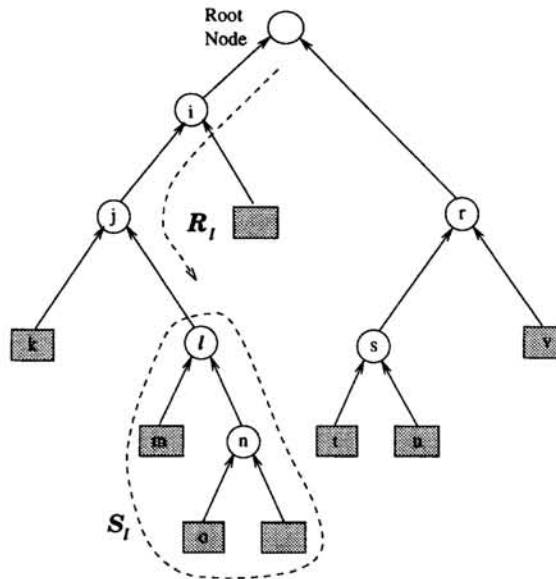

Figure 5: Path pruning in the HME.

### Path pruning simulations

Figure 6 shows the application of the pruning algorithm to the task of discriminating between two interlocking spirals. With no pruning the solution to the two-spirals takes over 4,000 CPU seconds, whereas with pruning the solution is achieved in 155 CPU seconds.

One problem which we encountered when implementing this algorithm was in computing updates for the parameters of the tree in the case of high pruning thresholds. If a node is visited too few times during a training pass, it will sometimes have too little data to form reliable statistics and thus the new parameter values may be unreliable and lead to instability. This is particularly likely when the gates are saturated. To avoid this saturation we use a simplified version of the regularisation scheme described in Waterhouse et al. (1995).

### CONCLUSIONS

We have presented two extensions to the standard HME architecture. By pruning branches either during training or evaluation we may significantly reduce the computational requirements of the HME. By applying tree growing we allow greater flexibility in the HME which results in faster training and more efficient use of parameters.

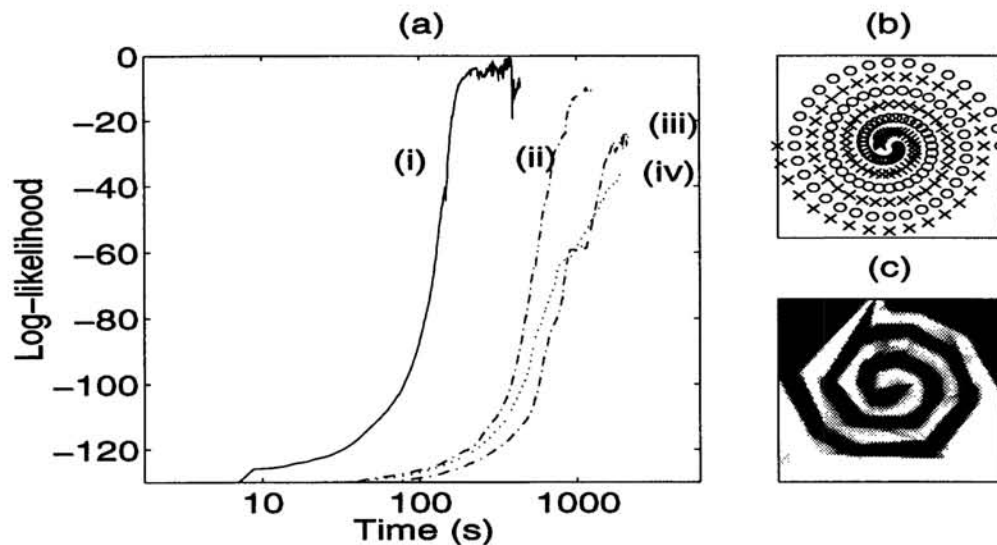

Figure 6: The effect of pruning on the two spirals classification problem by a 8 deep binary branching hme:(a) Log-likelihood vs. Time (CPU seconds), with log pruning thresholds for experts and gates $f$: (i) $f = -5.6$,(ii) $f = -10$,(iii) $f = -15$,(iv) no pruning, (b) training set for two-spirals task; the two classes are indicated by crosses and circles, (c) Solution to two spirals problem.

# References

Breiman, L., Friedman, J., Olshen, R. & Stone, C. J. (1984), *Classification and Regression Trees*, Wadswoth and Brooks/Cole.

Cun, Y. L., Denker, J. S. & Solla, S. A. (1990), Optimal brain damage, *in* D. S. Touretzky, ed., 'Advances in Neural Information Processing Systems 2', Morgan Kaufmann, pp. 598–605.

Dempster, A. P., Laird, N. M. & Rubin, D. B. (1977), 'Maximum likelihood from incomplete data via the EM algorithm', *Journal of the Royal Statistical Society, Series B* **39**, 1–38.

Fahlman, S. E. & Lebiere, C. (1990), The Cascade-Correlation learning architecture, Technical Report CMU-CS-90-100, School of Computer Science, Carnegie Mellon University, Pittsburgh, PA 15213.

Jordan, M. I. & Jacobs, R. A. (1994), 'Hierarchical Mixtures of Experts and the EM algorithm', *Neural Computation* **6**, 181–214.

Moody, J. E. (1992), The *effective* number of parameters: An analysis of generalization and regularization in nonlinear learning systems, *in* J. E. Moody, S. J. Hanson & R. P. Lippmann, eds, 'Advances in Neural Information Processing Systems 4', Morgan Kaufmann, San Mateo, California, pp. 847–854.

Waterhouse, S. R. & Robinson, A. J. (1994), Classification using hierarchical mixtures of experts, *in* 'IEEE Workshop on Neural Networks for Signal Processing', pp. 177–186.

Waterhouse, S. R., MacKay, D. J. C. & Robinson, A. J. (1995), Bayesian methods for mixtures of experts, *in* M. C. M. D. S. Touretzky & M. E. Hasselmo, eds, 'Advances in Neural Information Processing Systems 8', MIT Press.

Wolpert, D. H. (1993), Stacked generalization, Technical Report LA-UR-90-3460, The Santa Fe Institute, 1660 Old Pecos Trail, Suite A, Santa Fe, NM, 87501.
